# Active Comparison of Prediction Models

**Christoph Sawade, Niels Landwehr, and Tobias Scheffer**
University of Potsdam
Department of Computer Science
August-Bebel-Strasse 89, 14482 Potsdam, Germany
{sawade, landwehr, scheffer}@cs.uni-potsdam.de

## Abstract

We address the problem of comparing the risks of two given predictive models—for instance, a baseline model and a challenger—as confidently as possible on a fixed labeling budget. This problem occurs whenever models cannot be compared on held-out training data, possibly because the training data are unavailable or do not reflect the desired test distribution. In this case, new test instances have to be drawn and labeled at a cost. We devise an *active comparison* method that selects instances according to an instrumental sampling distribution. We derive the sampling distribution that maximizes the power of a statistical test applied to the observed empirical risks, and thereby minimizes the likelihood of choosing the inferior model. Empirically, we investigate model selection problems on several classification and regression tasks and study the accuracy of the resulting $p$-values.

## 1 Introduction

We address situations in which an informed choice between candidate predictive models—for instance, a baseline method and a challenger—has to be made. In practice, it is not always possible to compare the models' risks on held-out training data. For example, in computer vision it is common to acquire pre-trained object or face recognizers from third parties. Such recognizers do not typically come with the image databases that have been used to train them. The suppliers of the models could provide risk estimates based on held-out training data; however, such estimates might be biased because the training data would not necessarily reflect the distribution of images the deployed models will be exposed to. Another example are domains where the input distribution changes over a period of time in which a baseline model, *e.g.,* a spam filter, has been employed. By the time a new predictive model is considered, a previous risk estimate of the baseline model may no longer be accurate.

In these example scenarios, new test data have to be drawn and labeled. The standard approach to comparing models would be to draw $n$ test instances according to the test distribution which the model is exposed to in practice, label these data, and calculate the difference of the empirical risks $\hat{\Delta}_n$ and the sample variance $S_n^2$. Then, under the *null hypothesis* of identical risks, $\sqrt{n}\frac{\hat{\Delta}_n}{S_n}$ is asymptotically governed by a standard normal distribution, and we can compute a $p$-value which quantifies the likelihood that an observed empirical difference is due to chance, indicating how confidently the decision to prefer the apparently better model can be made.

In many application scenarios, unlabeled test instances are readily available whereas the process of labeling data is costly. We study an *active model comparison* process that, in analogy to active learning, selects instances from a pool of unlabeled test data and queries their labels. Instances are selected according to an instrumental sampling distribution $q$. The empirical difference of the models' risks is weighted appropriately to compensate for the discrepancy between instrumental and test distributions which leads to consistent—that is, asymptotically unbiased—risk estimates.

The principal theoretical contribution of this paper is the derivation of a sampling distribution $q$ that allows us to make the decision to prefer the superior model as confidently as possible given a fixed labeling budget $n$, if one of the models is in fact superior. Equivalently, one may use $q$ to minimize the labeling costs $n$ required to reach a correct decision at a prescribed level of confidence.

The active comparison problem that we study can be seen as an extreme case of active learning, in which the model space contains only two (or, more generally, a small number of) models. For the special case of classification with zero-one loss and two models under study, a simplified version of the sampling distribution we derive coincides with the sampling distribution used in the $A^2$ and *IWAL* active learning algorithms proposed by Balcan et al. [1] and Beygelzimer et al. [2]. For $A^2$ and *IWAL*, the derivation of this distribution is based on finite-sample complexity bounds, while in our approach, it is based on maximizing the power of a statistical test comparing the models under study. The latter approach has the advantage that it directly generalizes to regression problems. A further difference to active learning is that our goal is not only to choose the best model, but also to obtain a well-calibrated $p$-value indicating the confidence with which this decision can be made.

Our method is also related to recent work on active data acquisition strategies for the evaluation of a single predictive model, in terms of standard risks [8] or generalized risks that subsume precision, recall, and f-measure [9]. The problem addressed in this paper is different in that we seek to assess the *relative* performance of two models, without necessarily determining *absolute* risks precisely. Madani et al. have studied *active model selection*, where the goal is also to identify a model with lowest risk [5]. However, in their setting costs are associated with obtaining predictions $\hat{y} = f(\mathbf{x})$, while in our setting costs are associated with obtaining labels $y \sim p(y|\mathbf{x})$. *Hoeffding races* [6] and *sequential sampling algorithms* [10] perform efficient model selection by keeping track of risk bounds for candidate models and removing models that are clearly outperformed from consideration. The goal of these methods is to reduce computational complexity, not labeling effort.

The rest of this paper is organized as follows. The problem setting is laid out in Section 2. Section 3 derives the instrumental distribution and details our theoretical findings. Section 4 explores active model comparison experimentally. Section 5 concludes.

## 2   Problem Setting

Let $\mathcal{X}$ denote the feature space and $\mathcal{Y}$ the label space; an unknown test distribution $p(\mathbf{x}, y)$ is defined over $\mathcal{X} \times \mathcal{Y}$. Let $p(y|\mathbf{x}; \theta_1)$ and $p(y|\mathbf{x}; \theta_2)$ be given $\theta$-parameterized models of $p(y|\mathbf{x})$ and let $f_j : \mathcal{X} \to \mathcal{Y}$ with $f_j(\mathbf{x}) = \arg\max_y p(y|\mathbf{x}; \theta_j)$ be the corresponding predictive functions.

The risks of $f_1, f_2$ are given by

$$R[f_j] = \iint \ell(f_j(\mathbf{x}), y)p(\mathbf{x}, y)\mathrm{d}y\,\mathrm{d}\mathbf{x} \qquad (1)$$

for a loss function $\ell : \mathcal{Y} \times \mathcal{Y} \to \mathbb{R}$. In a classification setting, the integral over $\mathcal{Y}$ reduces to a sum. The standard approach to comparing models is to compare empirical risk estimates

$$\hat{R}_n[f_j] = \frac{1}{n}\sum_{i=1}^{n} \ell(f_j(\mathbf{x}_i), y_i), \qquad (2)$$

where $n$ test instances $(\mathbf{x}_i, y_i)$ are drawn from $p(\mathbf{x}, y) = p(\mathbf{x})p(y|\mathbf{x})$. We assume that unlabeled data are readily available, but acquiring labels $y$ for selected instances $\mathbf{x}$ according to $p(y|\mathbf{x})$ is a costly process that may involve a query to a human labeler.

Test instances need not necessarily be drawn according to the input distribution $p(\mathbf{x})$. We will focus on a data labeling process that draws test instances according to an instrumental distribution $q(\mathbf{x})$ rather than $p(\mathbf{x})$. Intuitively, $q(\mathbf{x})$ should be designed such as to prefer instances that highlight differences between the models $f_1$ and $f_2$. Let $q(\mathbf{x})$ denote an instrumental distribution with the property that $p(\mathbf{x}) > 0$ implies $q(\mathbf{x}) > 0$ for all $\mathbf{x} \in \mathcal{X}$. A consistent risk estimate is then given by

$$\hat{R}_{n,q}[f_j] = \frac{1}{W}\sum_{i=1}^{n} \frac{p(\mathbf{x}_i)}{q(\mathbf{x}_i)}\ell(f_j(\mathbf{x}_i), y_i), \qquad (3)$$

where $(\mathbf{x}_i, y_i) \sim q(\mathbf{x})p(y|\mathbf{x})$ and $W = \sum_{i=1}^{n} \frac{p(\mathbf{x}_i)}{q(\mathbf{x}_i)}$. Weighting factors $\frac{p(\mathbf{x}_i)}{q(\mathbf{x}_i)}$ compensate for the discrepancy between test and instrumental distribution, and the normalizer is the sum of weights. Because of the weighting factors, Equation 3 defines a consistent risk estimate (see [4], Chapter 2). Consistency means that the expected value of $\hat{R}_{n,q}[f_j]$ converges to the true risk $R[f_j]$ for $n \to \infty$.

Given estimates $\hat{R}_{n,q}[f_1]$ and $\hat{R}_{n,q}[f_2]$, the difference $\hat{\Delta}_{n,q} = \hat{R}_{n,q}[f_1] - \hat{R}_{n,q}[f_2]$ provides evidence on which model is preferable; a positive $\hat{\Delta}_{n,q}$ argues in favor of $f_2$. In preferring one model over the other, one rejects the *null hypothesis* that the observed difference $\hat{\Delta}_{n,q}$ is only a random effect, and $R[f_1] = R[f_2]$ holds. The null hypothesis implies that the mean of $\hat{\Delta}_{n,q}$ is asymptotically zero. Because $\hat{\Delta}_{n,q}$ is asymptotically normally distributed (see, *e.g.,* [3]), it further implies that the statistic

$$\sqrt{n}\frac{\hat{\Delta}_{n,q}}{\sigma_{n,q}} \sim \mathcal{N}(0,1)$$

is asymptotically standard-normally distributed, where $\frac{1}{n}\sigma_{n,q}^2 = \mathrm{Var}[\hat{\Delta}_{n,q}]$ denotes the variance of $\hat{\Delta}_{n,q}$. In practice, $\sigma_{n,q}^2$ is unknown. A consistent estimator of $\sigma_{n,q}^2$ is given by

$$S_{n,q}^2 = \frac{1}{W}\sum_{i=1}^{n}\frac{p(\mathbf{x}_i)^2}{q(\mathbf{x}_i)^2}\left(\ell(f_1(\mathbf{x}_i), y_i) - \ell(f_2(\mathbf{x}_i), y_i) - \hat{\Delta}_{n,q}\right)^2, \tag{4}$$

as shown, for example, by Geweke [3]. Substituting the empirical for the true standard deviation yields an observable statistic $\sqrt{n}\frac{\hat{\Delta}_{n,q}}{S_{n,q}}$. Because $S_{n,q}^2$ consistently estimates $\sigma_{n,q}^2$, the null hypothesis also implies that the observable statistic is asymptotically standard normally distributed,

$$\sqrt{n}\frac{\hat{\Delta}_{n,q}}{S_{n,q}} \sim \mathcal{N}(0,1).$$

Let $\Phi$ denote the cumulative distribution function of the standard normal distribution. Then,

$$2\left(1 - \Phi\left(\sqrt{n}\frac{|\hat{\Delta}_{n,q}|}{S_{n,q}}\right)\right) \tag{5}$$

is called the $p$-value of a *two-sided paired Wald test* (see, *e.g.,* [12], Chapter 10). The $p$-value quantifies the likelihood of observing the given absolute value of the test statistic, or a higher value, by chance under the null hypothesis. Student's $t$-distribution can serve as a more popular approximation of the distribution of a test statistic under the null hypothesis, resulting in the common $t$-*test*. Note, however, that $S_{n,q}$ would have to be a sum of squared, normally distributed random variables for the test statistic to be asymptotically governed by the $t$-distribution. This assumption is reasonable for regression, but not for classification, and only for the case of $p = q$.

If the null hypothesis does not hold and the two models incur different risks, the distribution of the test statistic depends on the chosen sampling distribution $q(\mathbf{x})$. Our goal is to find a distribution $q(\mathbf{x})$ that allows us to tell the risks of $f_1$ and $f_2$ apart with high confidence. More formally, the power of a test when sampling from $q(\mathbf{x})$ is the likelihood that the null hypothesis can be rejected, that is, the likelihood that the $p$-value falls below a pre-specified confidence threshold $\alpha$. Our goal is to find the sampling distribution $q$ that maximizes test power:

$$q^* = \arg\max_{q} p\left(2\left(1 - \Phi\left(\sqrt{n}\frac{|\hat{\Delta}_{n,q}|}{S_{n,q}}\right)\right) \le \alpha\right). \tag{6}$$

## 3 Active Model Comparison

We now turn towards deriving an optimal sampling distribution $q^*$ according to Equation 6. Section 3.1 analytically derives an asymptotically optimal sampling distribution. Section 3.2 discusses the sampling distribution in a pool-based setting and presents the active comparison algorithm.

## 3.1 Asymptotically Optimal Sampling

Let $\Delta = R[f_1] - R[f_2]$ denote the true risk difference, and assume $\Delta \neq 0$. Given a confidence threshold $\alpha$, the test power equals the probability that the absolute value of the test statistic exceeds the corresponding critical value $z_\alpha = \Phi^{-1}\left(1 - \frac{\alpha}{2}\right)$:

$$p\left(2 - 2\Phi\left(\sqrt{n}\frac{|\hat{\Delta}_{n,q}|}{S_{n,q}}\right) \leq \alpha\right) = p\left(\sqrt{n}\frac{|\hat{\Delta}_{n,q}|}{S_{n,q}} \geq z_\alpha\right). \tag{7}$$

Asymptotically, it holds that

$$\frac{\sqrt{n}(\hat{\Delta}_{n,q} - \Delta)}{\sigma_{n,q}} \sim \mathcal{N}(0,1).$$

Since $S_{n,q}$ consistently estimates $\sigma_{n,q}$, it follows that for large $n$ the statistic $\sqrt{n}\frac{\hat{\Delta}_{n,q}}{S_{n,q}}$ is normally distributed with mean $\frac{\sqrt{n}\Delta}{\sigma_{n,q}}$ and unit variance,

$$\frac{\sqrt{n}\hat{\Delta}_{n,q}}{S_{n,q}} \sim \mathcal{N}\left(\frac{\sqrt{n}\Delta}{\sigma_{n,q}}, 1\right). \tag{8}$$

Equation 8 implies that the absolute value $\sqrt{n}\frac{|\hat{\Delta}_{n,q}|}{S_{n,q}}$ of the test statistic follows a folded normal distribution with location parameter $\frac{\sqrt{n}\Delta}{\sigma_{n,q}}$ and scale parameter one. According to Equation 7, test power can thus be approximated in terms of the cumulative distribution of this folded normal distribution,

$$p\left(2 - 2\Phi\left(\sqrt{n}\frac{|\hat{\Delta}_{n,q}|}{S_{n,q}}\right) \leq \alpha\right) \approx 1 - \int_0^{z_\alpha} f\left(T; \frac{\sqrt{n}\Delta}{\sigma_{n,q}}, 1\right) dT, \tag{9}$$

where

$$f(T; \mu, 1) = \frac{1}{\sqrt{2\pi}}\exp\left(-\frac{1}{2}(T+\mu)^2\right) + \frac{1}{\sqrt{2\pi}}\exp\left(-\frac{1}{2}(T-\mu)^2\right)$$

denotes the density of a folded normal distribution with location parameter $\mu$ and scale parameter one. We define the shorthand

$$\beta_{n,q} = 1 - \int_0^{z_\alpha} f\left(T; \frac{\sqrt{n}\Delta}{\sigma_{n,q}}, 1\right) dT$$

for the approximation of test power given by Equation 9. In the following, we derive a sampling distribution maximizing $\beta_{n,q}$, thereby approximately solving the optimization problem of Equation 6.

**Theorem 1** (Optimal Sampling Distribution). *Let $\Delta = R[f_1] - R[f_2]$ with $\Delta \neq 0$. The distribution*

$$q^*(\mathbf{x}) \propto p(\mathbf{x})\sqrt{\int (\ell(f_1(\mathbf{x}), y) - \ell(f_2(\mathbf{x}), y) - \Delta)^2 \, p(y|\mathbf{x}) dy}$$

*asymptotically maximizes $\beta_{n,q}$; that is, for any other sampling distribution $q \neq q^*$ it holds that $\beta_{n,q} < \beta_{n,q^*}$ for sufficiently large $n$.*

Before we prove Theorem 1, we show that a sampling distribution asymptotically maximizes $\beta_{n,q}$ if and only if it minimizes the asymptotic variance of the estimator $\hat{\Delta}_{n,q}$.

**Lemma 2** (Variance Optimality). *Let $q$, $q'$ denote two sampling distributions. Then it holds that $\beta_{n,q} > \beta_{n,q'}$ for sufficiently large $n$ if and only if*

$$\lim_{n\to\infty} n\operatorname{Var}\left[\hat{\Delta}_{n,q}\right] < \lim_{n\to\infty} n\operatorname{Var}\left[\hat{\Delta}_{n,q'}\right]. \tag{10}$$

A proof is included in the online appendix. Lemma 2 shows that in order to solve the optimization problem given by Equation 6, we need to find the sampling distribution minimizing the asymptotic variance of the estimator $\hat{\Delta}_{n,q}$. This asymptotic variance is characterized by the following Lemma.

**Lemma 3** (Asymptotic Variance). *The asymptotic variance $\sigma_q^2 = \lim\limits_{n\to\infty} n\,\mathrm{Var}[\hat{\Delta}_{n,q}]$ of $\hat{\Delta}_{n,q}$ is given by*

$$\sigma_q^2 = \iint \frac{p(\mathbf{x})^2}{q(\mathbf{x})^2} \left(\ell(f_1(\mathbf{x}),y) - \ell(f_2(\mathbf{x}),y) - \Delta\right)^2 p(y|\mathbf{x})q(\mathbf{x})\mathrm{d}y\,\mathrm{d}\mathbf{x}.$$

A proof of Lemma 3 is included in the online appendix.

*Proof of Theorem 1.* We can now prove Theorem 1 by deriving the distribution $q^*$ that minimizes the asymptotic variance $\sigma_q^2$ as given by Lemma 3. We minimize the functional $\sigma_q^2$ in terms of $q$ under the constraint $\int q(\mathbf{x})\mathrm{d}\mathbf{x} = 1$ using a Lagrange multiplier $\beta$.

$$\mathcal{L}\left[q,\beta\right] = \sigma_q^2 + \beta\left(\int q(\mathbf{x})d\mathbf{x} - 1\right) = \int \frac{c(\mathbf{x})}{q(\mathbf{x})} + \beta\left(q(\mathbf{x}) - p(\mathbf{x})\right)\mathrm{d}\mathbf{x}$$

where $c(\mathbf{x}) = p(\mathbf{x})^2 \int \left(\ell(f_1(\mathbf{x}),y) - \ell(f_2(\mathbf{x}),y) - \Delta\right)^2 p(y|\mathbf{x})\mathrm{d}y$. The optimal point for the constrained problem satisfies the Euler-Lagrange equation

$$\frac{\partial}{\partial q(\mathbf{x})}\left(\frac{c(\mathbf{x})}{q(\mathbf{x})} + \beta\left(q(\mathbf{x}) - p(\mathbf{x})\right)\right) = -\frac{c(\mathbf{x})}{q(\mathbf{x})^2} + \beta = 0. \tag{11}$$

A solution for Equation 11 with respect to the normalization constraint is given by

$$q^*(\mathbf{x}) = \frac{\sqrt{c(\mathbf{x})}}{\int \sqrt{c(\mathbf{x})}\mathrm{d}\mathbf{x}}. \tag{12}$$

Resubstitution of $c(\mathbf{x})$ into Equation 12 implies the theorem. □

## 3.2 Empirical Sampling Distribution

The distribution $q^*$ also depends on the true conditional $p(y|\mathbf{x})$ and the true difference in risks $\Delta$. In order to implement the method, we have to approximate these quantities. Note that as long as $p(\mathbf{x}) > 0$ implies $q(\mathbf{x}) > 0$, any choice of $q$ will yield consistent risk estimates because weighting factors account for the discrepancy between sampling and test distribution (Equation 3). That is, $\hat{\Delta}_{n,q}$ is guaranteed to converge to $\Delta$ as $n$ grows large; any approximation employed to compute $q^*$ will only affect the number of test examples required to reach a certain level of estimation accuracy. To approximate the true conditional $p(y|\mathbf{x})$, we use the given predictive models $p(y|\mathbf{x}; \theta_1)$ and $p(y|\mathbf{x}; \theta_2)$, and assume a mixture distribution giving equal weight to both models:

$$p(y|\mathbf{x}) \approx \frac{1}{2}p(y|\mathbf{x}; \theta_1) + \frac{1}{2}p(y|\mathbf{x}; \theta_2). \tag{13}$$

The risk difference $\Delta$ is replaced by a difference $\Delta_\theta$ of introspective risks calculated from Equation 1, where the integral over $\mathcal{X}$ is replaced by a sum over the pool, $p(\mathbf{x}) = \frac{1}{m}$, and $p(y|\mathbf{x})$ is approximated by Equation 13.

We will now derive the empirical sampling distribution for two standard loss functions.

**Derivation 4** (Sampling for Zero-one Loss). *Let $\ell$ be the zero-one loss for a binary prediction problem with label space $\mathcal{Y} = \{0, 1\}$. When $p(y|\mathbf{x})$ is approximated as in Equation 13, the sampling distribution asymptotically maximizing $\beta_{n,q}$ in a pool-based setting resolves to*

$$q^*(\mathbf{x}) \propto \begin{cases} |\Delta_\theta| & : f_1(\mathbf{x}) = f_2(\mathbf{x}) \\ \sqrt{1 - 2\Delta_\theta(1 - 2p(y=1|\mathbf{x}; \theta)) + \Delta_\theta^2} & : f_1(\mathbf{x}) > f_2(\mathbf{x}) \\ \sqrt{1 + 2\Delta_\theta(1 - 2p(y=1|\mathbf{x}; \theta)) + \Delta_\theta^2} & : f_1(\mathbf{x}) < f_2(\mathbf{x}) \end{cases}$$

*for all $\mathbf{x} \in D$.*

A proof is included in the online appendix. Instead of using Approximation 13, an uninformative approximation $p(y=1|\mathbf{x}) \approx 0.5$ may be used. In this case $q^*$ degenerates to uniform sampling from the subset of the pool where $f_1(\mathbf{x}) \neq f_2(\mathbf{x})$. We denote this baseline as *active$_{\neq}$*. This baseline coincides with the $A^2$ as well as the *IWAL* active learning algorithms, applied to the model space $\{f_1, f_2\}$, as can be seen from inspection of Algorithm 1 in [1] and Algorithms 1 and 2 in [2].

We now derive the optimal sampling distribution for regression problems with a squared loss function, assuming that the predictive distributions $p(y|\mathbf{x}; \theta_1)$ and $p(y|\mathbf{x}; \theta_2)$ are Gaussian:

---

**Algorithm 1** Active Model Comparison

---

**input** Models $f_1$, $f_2$ with distributions $p(y|\mathbf{x}; \theta_1)$, $p(y|\mathbf{x}; \theta_2)$; pool $D$; labeling budget $n$.
 1: Compute sampling distribution $q^*$ (Derivation 4 or 5).
 2: **for** $i = 1, \ldots, n$ **do**
 3:     Draw $\mathbf{x}_i \sim q^*(\mathbf{x})$ from $D$ with replacement.
 4:     Query label $y_i \sim p(y|\mathbf{x}_i)$ from oracle.
 5: **end for**
 6: Compute $\hat{R}_{n,q}[f_1]$ and $\hat{R}_{n,q}[f_2]$ (Equation 3).
 7: Determine $f^* \leftarrow \arg\min_{f \in \{f_1, f_2\}} \hat{R}_{n,q}[f]$, compute $p$-value for sample (Equation 5)
**output** $f^*$, $p$-value.

---

**Derivation 5** (Sampling for Squared Loss). *Let $\ell$ be the squared loss, and let $p(y|\mathbf{x}; \theta_1)$ and $p(y|\mathbf{x}; \theta_2)$ be Gaussian. When $p(y|\mathbf{x})$ is approximated as in Equation 13, then the sampling distribution asymptotically maximizing $\beta_{n,q}$ in a pool-based setting resolves to*

$$q^*(\mathbf{x}) \propto \sqrt{2\left(f_1(\mathbf{x}) - f_2(\mathbf{x})\right)^2 \left(f_1^2(\mathbf{x}) + f_2^2(\mathbf{x}) + \tau_{\mathbf{x}}^2\right) - \left(f_1^2(\mathbf{x}) - f_2^2(\mathbf{x})\right)^2} \tag{14}$$

*for all $\mathbf{x} \in D$, where $\tau_{\mathbf{x}}^2$ denotes the sum of the variances of the predictive distributions at $\mathbf{x} \in D$.*

A proof is given in the online appendix. Variances of predictive distributions at instance $\mathbf{x}$ would be available from a probabilistic model such as a Gaussian process [7]. If only predictions $f_j(\mathbf{x})$ but no predictive distribution is available, we can assume peaked distributions with $\tau_{\mathbf{x}}^2 \to 0$, leading to

$$q^*(\mathbf{x}) \propto (f_1(\mathbf{x}) - f_2(\mathbf{x}))^2,$$

or we can assume infinitely broad predictive distributions with $\tau_{\mathbf{x}}^2 \to \infty$, leading to

$$q^*(\mathbf{x}) \propto |f_1(\mathbf{x}) - f_2(\mathbf{x})|.$$

We refer to these baselines as *active$_0$* and *active$_\infty$*.

Algorithm 1 summarizes the active model comparison algorithm. It samples $n$ instances with replacement from the pool according to the distribution prescribed by Derivations 4 (for zero-one loss) or 5 (for squared loss) and queries their label. Note that instances can be drawn more than once; in the special case that the labeling process is deterministic, the actual labeling costs may thus stay below the sample size. In this case, the loop is continued until the labeling budget is exhausted.

We have so far focused on the problem of comparing the risks of two prediction models, such as a baseline and a challenger. We might also consider several alternative models; the objective of an evaluation could be to rank the models according to the risk incurred or to identify the model with lowest risk. Standard generalizations of the Wald test that compare multiple alternatives—for instance, within-subject ANOVA [11]—try to reject the null hypothesis that the means of all considered alternatives are equal. Rejection does not imply that all empirically observed differences are significant; for instance, the test could become significant because one of the alternatives performs clearly worst. Choosing a sampling distribution $q$ that maximizes the power of such a test would thus in general not reflect the objectives of the empirical evaluation.

In practice, researchers often resort to pairwise hypothesis testing when comparing multiple prediction models. Accordingly, we derive a heuristic sampling distribution for the comparison of multiple models $\theta_1, \ldots, \theta_k$ as a mixture of pairwise-optimal sampling distributions,

$$q^*(\mathbf{x}) = \frac{1}{k(k-1)} \sum_{i \neq j} q_{i,j}^*(\mathbf{x}), \tag{15}$$

where $q_{i,j}^*$ denotes the optimal distribution for comparing the models $\theta_i$ and $\theta_j$ given by Theorem 1. When comparing multiple models, we replace Equation 13 by a mixture over all models $\theta_1, \ldots, \theta_k$.

## 4   Empirical Results

We study the empirical behavior of active comparison (Algorithm 1, labeled *active* in all diagrams) relative to a risk comparison based on a test sample drawn uniformly from the pool (labeled *passive*)

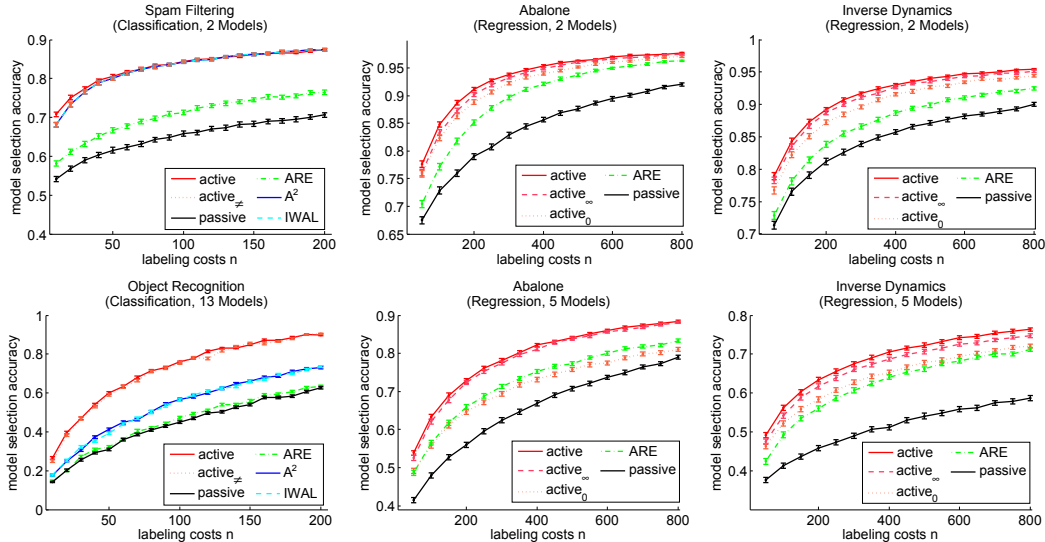

Figure 1: Model selection accuracy over labeling costs for comparison of two prediction models (top) and multiple prediction models (bottom). Error bars indicate the standard error.

and the baselines $active_{\neq}$, $active_0$, and $active_{\infty}$ discussed in Section 3.2. We also include the active risk estimator presented in [8] in our study, which infers optimal sampling distributions $q_1^*$ and $q_2^*$ for individually estimating the risks of the models $\theta_1$ and $\theta_2$. Test instances are sampled from a mixture distribution $q^*(\mathbf{x}) = \frac{1}{2}q_1^*(\mathbf{x}) + \frac{1}{2}q_2^*(\mathbf{x})$ (labeled *ARE*). Each comparison method returns the model with lower empirical risk and the $p$-value of a paired two-sided test. When studying classification, we also include the active learning algorithms $A^2$ [1] and *IWAL* [2] as baselines by using them to sample test instances. Their model space is the set of predictive models that are to be compared.

We conduct experiments in two classification domains (spam filtering, object recognition) and two regression domains (inverse dynamics, Abalone) ranging from 4,109 to 169,612 instances. Kernelized logistic regression is employed for classification, Gaussian processes are employed for regression. In the spam filtering domain, we compare models that differ in the recency of their training data. In the object recognition domain, we compare SIFT-based recognizers using different interest point detectors (Harris operator, Canny edge detector, Förstner operator) and visual vocabularies. For regression, we compare models that differ in the choice of their kernel function (linear versus Matern, polynomial kernels of different degrees). Models are trained on part of the available data; the rest of the data serve as the pool of unlabeled test instances for which labels can be queried. Results are averaged over 5,000 repetitions of the evaluation process. Further details on the datasets and experimental setup are included in the online appendix.

## 4.1 Identifying the Model With Lower True Risk

We measure *model selection accuracy*, defined as the fraction of experiments in which an evaluation method correctly identifies the model with lower true risk. The true risk is taken to be the risk over all test instances in the pool. Figure 1 (top) shows that for the comparison of two models *active* results in significantly higher model selection accuracy than *passive*, or, equivalently, saves between 70% and 90% of labeling effort. Differences between *active* and the simplified variants $active_0$, $active_{\infty}$, and $active_{\neq}$ are marginal. These variants do not require an estimate of $p(y|\mathbf{x})$, thus the method is applicable even if no such estimate is available. $A^2$ and *IWAL* coincide with $active_{\neq}$ (cf. Section 3.2). Figure 1 (bottom) shows results when comparing multiple models. In the object recognition domain, *active* saves approximately 70% of labeling effort compared to *passive*. $A^2$ and *IWAL* outperform *passive* but are less accurate than *active*. For the regression domains, *active* saves between 60% and 85% of labeling effort compared to *passive*.

## 4.2 Significance Testing: Type I and Type II Errors

We now study how often a comparison method is able to reject the null hypothesis that two predictive models incur identical risks, and the calibration of the resulting $p$-values. For classification, the

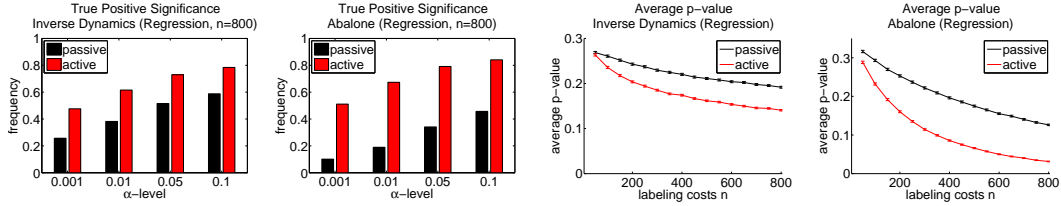

Figure 2: True-positive significance rate for different test levels $\alpha$ (left, left-center). Average $p$-value over labeling costs $n$ (right-center, right). Error bars indicate the standard error.

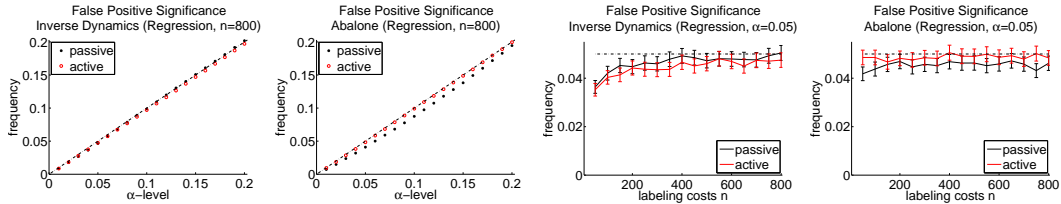

Figure 3: False-positive significance rate over test level $\alpha$ (left, left-center). False-positive significance rate over labeling costs $n$ (right-center, right). Error bars indicate the standard error.

method $active_{\neq}$ is equivalent to *passive* applied to $D_{\neq} = \{\mathbf{x} \in D | f_1(\mathbf{x}) \neq f_2(\mathbf{x})\}$ (see Section 3.2). Labeling effort is thus simply reduced by a factor of $|D_{\neq}|/|D|$. For regression, the analysis is less straightforward as typically $D = D_{\neq}$. In this section we therefore focus on regression problems.

Figure 2 (left, left-center) shows how often the active and passive comparison methods are able to reject the null hypothesis that the two models incur identical risk. The true risks incurred are never equal in these experiments. We observe that *active* is able to reject the null hypothesis more often and with a higher confidence. In the Abalone domain, *active* rejects the null hypothesis at $\alpha = 0.001$ more often than *passive* is able to reject it at $\alpha = 0.1$. Figure 2 (right-center, right) shows that active comparison also results in lower average $p$-values, in particular for large $n$.

We also conduct experiments under the null hypothesis. Whenever a test instance $\mathbf{x}$ is sampled and the predictions $y = f_1(\mathbf{x})$ and $y' = f_2(\mathbf{x})$ are queried, the predicted labels $y$ and $y'$ are swapped with probability 0.5; this ensures that the true risks of $f_1$ and $f_2$ coincide. Figure 3 (left, left-center) shows that Type I errors are well calibrated for both tests, as the false-positive rate stays below the (ideal) diagonal line when plotted against $\alpha$. Figure 3 (right-center, right) shows that both tests are slightly conservative for small $n$, and approach the expected false-positive rate as $n$ grows larger.

We finally study a protocol in which test instances are drawn and labeled until the null hypothesis can be rejected or the labeling budget is exhausted. Results (included in the online appendix) indicate that *active* incurs the lowest average labeling costs, obtains significance results most often, and has the lowest likelihood of incorrectly choosing the model with higher true risk.

## 5  Conclusion

We have derived the sampling distribution that asymptotically maximizes the power of a statistical test that compares the risk of two predictive models. The sampling distribution intuitively gives preference to test instances on which the models disagree strongly.

Empirically, we observed that the resulting active comparison method consistently outperforms a traditional comparison based on a uniform sample of test instances. Active comparison identifies the model with lower true risk more often, and is able to detect significant differences between the risks of two given models more quickly. In the four experimental domains that we studied, performing active comparison resulted in a saved labeling effort of between 60% and over 90%. We also performed experiments under the null hypothesis that both models incur identical risks, and verified that active comparison does not lead to increased false-positive significance results.

### Acknowledgements

We wish to thank Paul Prasse for his help with the experiments on object recognition data.

# References

[1] M. Balcan, A. Beygelzimer, and J. Langford. Agnostic active learning. In *Proceedings of the 23rd International Conference on Machine Learning*, 2006.

[2] A. Beygelzimer, S. Dasgupta, and J. Langford. Importance weighted active learning. In *Proceedings of the 26th International Conference on Machine Learning*, 2009.

[3] J. Geweke. Bayesian inference in econometric models using monte carlo integration. *Econometrica*, 57(6):1317–1339, 1989.

[4] J. S. Liu. *Monte Carlo Strategies in Scientific Computing*. Springer, 2001.

[5] O. Madani, D. J. Lizotte, and R. Greiner. Active model selection. In *Proceedings of the 20th Conference on Uncertainty in Artificial Intelligence*, 2004.

[6] O. Maron and A. W. Moore. Hoeffding races: Accelerating model selection search for classification and function approximation. In *Proceedings of the 6th Annual Conference on Neural Information Processing Systems*, 1993.

[7] Carl Edward Rasmussen and Christopher Williams. *Gaussian Processes for Machine Learning*. MIT Press, 2006.

[8] C. Sawade, N. Landwehr, S. Bickel, and T. Scheffer. Active risk estimation. In *Proceedings of the 27th International Conference on Machine Learning*, 2010.

[9] C. Sawade, N. Landwehr, and T. Scheffer. Active estimation of f-measures. In *Proceedings of the 23rd Annual Conference on Neural Information Processing Systems*, 2010.

[10] T. Scheffer and S. Wrobel. Finding the most interesting patterns in a database quickly by using sequential sampling. *Journal of Machine Learning Research*, 3:833–862, 2003.

[11] D. Sheskin. *Handbook of Parametric and Nonparametric Statistical Procedures*. Chapman & Hall, 2004.

[12] L. Wasserman. *All of Statistics: a Concise Course in Statistical Inference*. Springer, 2004.

